# FilterBoost: Regression and Classification on Large Datasets

**Joseph K. Bradley**
Machine Learning Department
Carnegie Mellon University
Pittsburgh, PA 15213
jkbradle@cs.cmu.edu

**Robert E. Schapire**
Department of Computer Science
Princeton University
Princeton, NJ 08540
schapire@cs.princeton.edu

## Abstract

We study boosting in the filtering setting, where the booster draws examples from an oracle instead of using a fixed training set and so may train efficiently on very large datasets. Our algorithm, which is based on a logistic regression technique proposed by Collins, Schapire, & Singer, requires fewer assumptions to achieve bounds equivalent to or better than previous work. Moreover, we give the first proof that the algorithm of Collins et al. is a strong PAC learner, albeit within the filtering setting. Our proofs demonstrate the algorithm's strong theoretical properties for both classification and conditional probability estimation, and we validate these results through extensive experiments. Empirically, our algorithm proves more robust to noise and overfitting than batch boosters in conditional probability estimation and proves competitive in classification.

## 1 Introduction

Boosting provides a ready method for improving existing learning algorithms for classification. Taking a weaker learner as input, boosters use the weak learner to generate weak hypotheses which are combined into a classification rule more accurate than the weak hypotheses themselves. Boosters such as AdaBoost [1] have shown considerable success in practice.

Most boosters are designed for the *batch* setting where the learner trains on a fixed example set. This setting is reasonable for many applications, yet it requires collecting all examples before training. Moreover, most batch boosters maintain distributions over the entire training set, making them computationally costly for very large datasets. To make boosting feasible on larger datasets, learners can be designed for the *filtering* setting. The batch setting provides the learner with a fixed training set, but the filtering setting provides an oracle which can produce an unlimited number of labeled examples, one at a time. This idealized model may describe learning problems with on-line example sources, including very large datasets which must be loaded piecemeal into memory. By using new training examples each round, filtering boosters avoid maintaining a distribution over a training set and so may use large datasets much more efficiently than batch boosters.

The first polynomial-time booster, by Schapire, was designed for filtering [2]. Later filtering boosters included two more efficient ones proposed by Freund, but both are non-adaptive, requiring a priori bounds on weak hypothesis error rates and combining weak hypotheses via unweighted majority votes [3,4]. Domingo & Watanabe's MadaBoost is competitive with AdaBoost empirically but theoretically requires weak hypotheses' error rates to be monotonically increasing, an assumption we found to be violated often in practice [5]. Bshouty & Gavinsky proposed another, but, like Freund's, their algorithm requires an a priori bound on weak hypothesis error rates [6]. Gavinsky's AdaFlat$_{filt}$ algorithm and Hatano's GiniBoost do not have these limitations, but the former has worse bounds than other adaptive algorithms while the latter explicitly requires finite weak hypothesis spaces [7,8].

This paper presents FilterBoost, an adaptive boosting-by-filtering algorithm. We show it is applicable to both conditional probability estimation, where the learner predicts the *probability* of each label given an example, and classification. In Section 2, we describe the algorithm, after which we interpret it as a stepwise method for fitting an additive logistic regression model for conditional probabilities. We then bound the number of rounds and examples required to achieve any target error in $(0, 1)$. These bounds match or improve upon those for previous filtering boosters but require fewer assumptions. We also show that FilterBoost can use the confidence-rated predictions from weak hypotheses described by Schapire & Singer [9].

In Section 3, we give results from extensive experiments. For conditional probability estimation, we show that FilterBoost often outperforms batch boosters, which prove less robust to overfitting. For classification, we show that filtering boosters' efficiency on large datasets allows them to achieve higher accuracies faster than batch boosters in many cases.

FilterBoost is based on a modification of AdaBoost by Collins, Schapire & Singer designed to minimize logistic loss [10]. Their batch algorithm has yet to be shown to achieve arbitrarily low test error, but we use techniques similar to those of MadaBoost to adapt the algorithm to the filtering setting and prove generalization bounds. The result is an adaptive algorithm with realistic assumptions and strong theoretical properties. Its robustness and efficiency on large datasets make it competitive with existing methods for both conditional probability estimation and classification.

## 2 The FilterBoost Algorithm

Let $X$ be the set of examples and $Y$ a discrete set of labels. For simplicity, assume $X$ is countable, and consider only binary labels $Y = \{-1, +1\}$. We assume there exists an unknown target distribution $D$ over labeled examples $(x, y) \in X \times Y$ from which training and test examples are generated. The goal in classification is to choose a hypothesis $h : X \to Y$ which minimizes the classification error $\Pr_D[h(x) \neq y]$, where the subscript indicates that the probability is with respect to $(x, y)$ sampled randomly from $D$.

In the batch setting, a booster is given a fixed training set $S$ and a weak learner which, given any distribution $D_t$ over training examples $S$, is guaranteed to return a weak hypothesis $h_t : X \to \mathbb{R}$ such that the error $\epsilon_t \equiv \Pr_{D_t}[\text{sign}(h_t(x)) \neq y] < 1/2$. For $T$ rounds $t$, the booster builds a distribution $D_t$ over $S$, runs the weak learner on $S$ and $D_t$, and receives $h_t$. The booster usually then estimates $\epsilon_t$ using $S$ and weights $h_t$ with $\alpha_t = \alpha_t(\epsilon_t)$. After $T$ rounds, the booster outputs a final hypothesis $H$ which is a linear combination of the weak hypotheses (e.g. $H(x) = \sum_t \alpha_t h_t(x)$). The sign of $H(x)$ indicates the predicted label $\hat{y}$ for $x$.

Two key elements of boosting are constructing $D_t$ over $S$ and weighting weak hypotheses. $D_t$ is built such that misclassified examples receive higher weights than in $D_{t-1}$, eventually forcing the weak learner to classify previously poorly classified examples correctly. Weak hypotheses $h_t$ are generally weighted such that hypotheses with lower errors receive higher weights.

### 2.1 Boosting-by-Filtering

We describe a general framework for boosting-by-filtering which includes most existing algorithms as well as our algorithm Filterboost. The filtering setting assumes the learner has access to an example oracle, allowing it to use entirely new examples sampled i.i.d. from $D$ on each round. However, while maintaining the distribution $D_t$ is straightforward in the batch setting, there is no fixed set $S$ on which to define $D_t$ in filtering. Instead, the booster simulates examples drawn from $D_t$ by drawing examples from $D$ via the oracle and reweighting them according to $D_t$. Filtering boosters generally accept each example $(x, y)$ from the oracle for training on round $t$ with probability proportional to the example's weight $D_t(x, y)$. The mechanism which accepts examples from the oracle with some probability is called the filter.

Thus, on each round, a boosting-by-filtering algorithm draws a set of examples from $D_t$ via the filter, trains the weak learner on this set, and receives a weak hypothesis $h_t$. Though a batch booster would estimate $\epsilon_t$ using the fixed set $S$, filtering boosters may use new examples from the filter. Like batch boosters, filtering boosters may weight $h_t$ using $\alpha_t = \alpha_t(\epsilon_t)$, and they output a linear combination of $h_1, \ldots, h_T$ as a final hypothesis.

The filtering setting allows the learner to estimate the error of $H_t$ to arbitrary precision by sampling from $D$ via the oracle, so FilterBoost does this to decide when to stop boosting.

## 2.2 FilterBoost

FilterBoost, given in Figure 1, is modeled after the aforementioned algorithm by Collins et al. [10] and Mada-Boost [5]. Given an example oracle, weak learner, target error $\varepsilon \in (0,1)$, and confidence parameter $\delta \in (0,1)$ upper-bounding the probability of failure, it iterates until the current combined hypothesis $H_t$ has error $\leq \varepsilon$.

On round $t$, FilterBoost draws $m_t$ examples from the filter to train the weak learner and get $h_t$. The number $m_t$ must be large enough to ensure $h_t$ has error $\epsilon_t < 1/2$ with high probability. The edge of $h_t$ is $\gamma_t = 1/2 - \epsilon_t$, and this edge is estimated by the function $getEdge()$, discussed below, and is used to set $h_t$'s weight $\alpha_t$. The current combined hypothesis is defined as $H_t = \text{sign}(\sum_{t'=1}^{t} \alpha_{t'} h_{t'})$.

The $Filter()$ function generates $(x,y)$ from $D_t$ by repeatedly drawing $(x,y)$ from the oracle, calculating the weight $q_t(x,y) \propto D_t(x,y)$, and accepting $(x,y)$ with probability $q_t(x,y)$.

Function $getEdge()$ uses a modification of the Nonmonotonic Adaptive Sampling method of Watanabe [11] and Domingo, Galvadà & Watanabe [12]. Their algorithm draws an adaptively chosen number of examples from the filter and returns an estimate $\hat{\gamma}_t$ of the edge of $h_t$ within relative error $\tau$ of the true edge $\gamma_t$ with high probability. The $getEdge()$ function revises this estimate as $\hat{\gamma}_t' = \hat{\gamma}_t/(1+\tau)$.

Define $F_t(x) \equiv \sum_{t'=1}^{t-1} \alpha_{t'} h_{t'}(x)$

Algorithm $FilterBoost$ accepts $Oracle(), \varepsilon, \delta, \tau$:

    For $t = 1, 2, 3, \dots$

        $\delta_t \longleftarrow \frac{\delta}{3t(t+1)}$

        Call $Filter(t, \delta_t, \varepsilon)$ to get $m_t$ examples to train WL; get $h_t$

        $\hat{\gamma}_t' \longleftarrow getEdge(t, \tau, \delta_t, \varepsilon)$

        $\alpha_t \longleftarrow \frac{1}{2} \ln\left(\frac{1/2 + \hat{\gamma}_t'}{1/2 - \hat{\gamma}_t'}\right)$

        Define $H_t(x) = \text{sign}\left(F_{t+1}(x)\right)$

    (Algorithm exits from $Filter()$ function.)

Function $Filter(t, \delta_t, \varepsilon)$ returns $(x,y)$

    Define $r = $ # calls to Filter so far on round $t$

    $\delta_t' \longleftarrow \frac{\delta_t}{r(r+1)}$

    For $(i = 0; i < \frac{2}{\varepsilon}\ln(\frac{1}{\delta_t'}); i = i+1)$:

        $(x,y) \longleftarrow Oracle()$

        $q_t(x,y) \longleftarrow \frac{1}{1 + e^{yF_t(x)}}$

        Return $(x,y)$ with probability $q_t(x,y)$

    End algorithm; return $H_{t-1}$

Function $getEdge(t, \tau, \delta_t, \varepsilon)$ returns $\hat{\gamma}_t'$

    Let $m \longleftarrow 0, n \longleftarrow 0, u \longleftarrow 0, \alpha \longleftarrow \infty$

    While $(|u| < \alpha(1 + 1/\tau))$:

        $(x,y) \longleftarrow Filter(t, \delta_t, \varepsilon)$

        $n \longleftarrow n + 1$

        $m \longleftarrow m + I(h_t(x) = y)$

        $u \longleftarrow m/n - 1/2$

        $\alpha \longleftarrow \sqrt{(1/2n)\ln(n(n+1)/\delta_t)}$

    Return $u/(1+\tau)$

Figure 1: The algorithm FilterBoost.

## 2.3 Analysis: Conditional Probability Estimation

We begin our analysis of FilterBoost by interpreting it as an additive model for logistic regression, for this interpretation will later aid in the analysis for classification. Such models take the form

$$\log \frac{\Pr[y=1|x]}{\Pr[y=-1|x]} = \sum_t f_t(x) = F(x), \qquad \text{which implies} \qquad \Pr[y=1|x] = \frac{1}{1 + e^{-F(x)}}$$

where, for FilterBoost, $f_t(x) = \alpha_t h_t(x)$. Dropping subscripts, we can write the expected negative log likelihood of example $(x,y)$ after round $t$ as

$$\pi(F_t + \alpha_t h_t) = \pi(F + \alpha h) = \text{E}\left[-\ln \frac{1}{1 + e^{-y(F(x) + \alpha h(x))}}\right] = \text{E}\left[\ln\left(1 + e^{-y(F(x) + \alpha h(x))}\right)\right].$$

Taking a similar approach to the analysis of AdaBoost in [13], we show in the following theorem that FilterBoost performs an approximate stepwise minimization of this negative log likelihood. The proof is in the Appendix.

**Theorem 1** *Define the expected negative log likelihood $\pi(F + \alpha h)$ as above. Given $F$, FilterBoost chooses $h$ to minimize a second-order Taylor expansion of $\pi$ around $h = 0$. Given this $h$, it then chooses $\alpha$ to minimize an upper bound of $\pi$.*

The batch booster given by Collins et al. [10] which FilterBoost is based upon is guaranteed to converge to the minimum of this objective when working over a finite sample. Note that Filter-Boost uses weak learners which are simple classifiers to perform regression. AdaBoost too may be interpreted as an additive logistic regression model of the form $\Pr[y = 1|x] = \frac{1}{1+e^{-2F(x)}}$ with $E[\exp(-yF(x))]$ as the optimization objective [13].

## 2.4 Analysis: Classification

In this section, we interpret FilterBoost as a traditional boosting algorithm for classification and prove bounds on its generalization error. We first give a theorem relating $err_t$, the error rate of $H_t$ over the target distribution $D$, to $p_t$, the probability with which the filter accepts a random example generated by the oracle on round $t$.

**Theorem 2** *Let $err_t = \Pr_D[H_t(x) \neq y]$, and let $p_t = E_D[q_t(x, y)]$. Then $err_t \leq 2p_t$.*

**Proof:**
$$
\begin{aligned}
err_t &= \Pr_D[H_t(x) \neq y] = \Pr_D[yF_{t-1}(x) \leq 0] \\
&= \Pr_D[q_t(x,y) \geq 1/2] \leq 2 \cdot E_D[q_t(x,y)] \\
&= 2p_t \quad \text{(using Markov's inequality above)} \qquad \blacksquare
\end{aligned}
$$

We next use the expected negative log likelihood $\pi$ from Section 2.3 as an auxiliary function to aid in bounding the required number of boosting rounds. Viewing $\pi$ as a function of the boosting round $t$, we can write $\pi_t = -\sum_{(x,y)} D(x,y) \ln(1 - q_t(x,y))$. Our goal is then to minimize $\pi_t$, and the following lemma captures the learner's progress in terms of the decrease in $\pi_t$ on each round. This lemma assumes edge estimates returned by $getEdge()$ are exact, i.e. $\hat{\gamma}_t' = \gamma_t$, which leads to a simpler bound on $T$ in Theorem 3. We then consider the error in edge estimates and give a revised bound in Lemma 2 and Theorem 5. The proofs of Lemmas 1 and 2 are in the Appendix.

**Lemma 1** *Assume for all $t$ that $\gamma_t \neq 0$ and $\gamma_t$ is estimated exactly. Let $\pi_t = -\sum_{(x,y)} D(x,y) \ln(1 - q_t(x,y))$. Then*

$$
\pi_t - \pi_{t+1} \geq p_t \left(1 - 2\sqrt{1/4 - \gamma_t^2}\right).
$$

Combining Theorem 2, which bounds the error of the current combined hypothesis in terms of $p_t$, with Lemma 1 gives the following upper bound on the required rounds $T$.

**Theorem 3** *Let $\gamma = \min_t |\gamma_t|$, and let $\varepsilon$ be the target error. Given Lemma 1's assumptions, if FilterBoost runs*

$$
T > \frac{2\ln(2)}{\varepsilon \left(1 - 2\sqrt{1/4 - \gamma^2}\right)}
$$

*rounds, then $err_t < \varepsilon$ for some $t$, $1 \leq t \leq T$. In particular, this is true for $T > \frac{\ln(2)}{2\varepsilon\gamma^2}$.*

**Proof:** For all $(x, y)$, since $F_1(x, y) = 0$, then $q_1(x, y) = 1/2$ and $\pi_1 = \ln(2)$. Now, suppose $err_t \geq \varepsilon, \forall t \in \{1, ..., T\}$. Then, from Theorem 2, $p_t \geq \varepsilon/2$, so Lemma 1 gives

$$
\pi_t - \pi_{t+1} \geq \frac{1}{2}\varepsilon\left(1 - 2\sqrt{1/4 - \gamma^2}\right)
$$

Unraveling this recursion as $\sum_{t=1}^{T}(\pi_t - \pi_{t+1}) = \pi_1 - \pi_{T+1} \leq \pi_1$ gives

$$
T \leq \frac{2\ln(2)}{\varepsilon\left(1 - 2\sqrt{1/4 - \gamma^2}\right)}.
$$

So, $err_t \geq \varepsilon, \forall t \in \{1, ..., T\}$ is contradicted if $T$ exceeds the theorem's lower bound. The simplified bound follows from the first bound via the inequality $1 - \sqrt{1-x} \leq x$ for $x \in [0,1]$. ∎

Theorem 3 shows FilterBoost can reduce generalization error to any $\varepsilon \in (0,1)$, but we have thus far overlooked the probabilities of failure introduced by three steps: training the weak learner, deciding when to stop boosting, and estimating edges. We bound the probability of each of these steps failing on round $t$ with a confidence parameter $\delta_t = \frac{\delta}{3t(t+1)}$ so that a simple union bound ensures the probability of some step failing to be at most FilterBoost's confidence parameter $\delta$. Finally, we revise Lemma 1 and Theorem 3 to account for error in estimating edges.

The number $m_t$ of examples the weak learner trains on must be large enough to ensure weak hypothesis $h_t$ has a non-zero edge and should be set according to the choice of weak learner.

To decide when to stop boosting (i.e. when $err_t \leq \varepsilon$), we can use Theorem 2, which upper-bounds the error of the current combined hypothesis $H_t$ in terms of the probability $p_t$ that $Filter()$ accepts a random example from the oracle. If the filter rejects enough examples in a single call, we know $p_t$ is small, so $H_t$ is accurate enough. Theorem 4 formalizes this intuition; the proof is in the Appendix.

**Theorem 4** *In a single call to $Filter(t)$, if $n$ examples have been rejected, where $n \geq \frac{2}{\varepsilon}\ln(1/\delta_t')$, then $err_t \leq \varepsilon$ with probability at least $1 - \delta_t'$.*

Theorem 4 provides a stopping condition which is checked on each call to $Filter()$. Each check may fail with probability at most $\delta_t' = \frac{\delta_t}{r(r+1)}$ on the $r^{\text{th}}$ call to $Filter()$ so that a union bound ensures FilterBoost stops prematurely on round $t$ with probability at most $\delta_t$. Theorem 4 uses a similar argument to that used for MadaBoost, giving similar stopping criteria for both algorithms.

We estimate weak hypotheses' edges $\gamma_t$ using the Nonmonotonic Adaptive Sampling (NAS) algorithm [11,12] used by MadaBoost. To compute an estimate $\hat{\gamma}_t$ of the true edge $\gamma_t$ within relative error $\tau \in (0,1)$ with probability $\geq 1 - \delta_t$, the NAS algorithm uses at most $\frac{2(1+2\tau)^2}{(\tau\gamma_t)^2}\ln(\frac{1}{\tau\gamma_t\delta_t})$ filtered examples. With this guarantee on edge estimates, we can rewrite Lemma 1 as follows:

**Lemma 2** *Assume for all $t$ that $\gamma_t \neq 0$ and $\gamma_t$ is estimated to within $\tau \in (0,1)$ relative error. Let $\pi_t = -\sum_{(x,y)} D(x,y)\ln(1 - q_t(x,y))$. Then*

$$\pi_t - \pi_{t+1} \geq p_t\left(1 - 2\sqrt{1/4 - \gamma_t^2\left(\frac{1-\tau}{1+\tau}\right)^2}\right).$$

Using Lemma 2, the following theorem modifies Theorem 3 to account for error in edge estimates.

**Theorem 5** *Let $\gamma = \min_t |\gamma_t|$. Let $\varepsilon$ be the target error. Given Lemma 2's assumptions, if FilterBoost runs*

$$T > \frac{2\ln(2)}{\varepsilon\left(1 - 2\sqrt{1/4 - \gamma^2(\frac{1-\tau}{1+\tau})^2}\right)}$$

*rounds, then $err_t < \varepsilon$ for some $t$, $1 \leq t \leq T$.*

The bounds from Theorems 3 and 5 show FilterBoost requires at most $O(\varepsilon^{-1}\gamma^{-2})$ boosting rounds. MadaBoost [5], which we test in our experiments, resembles FilterBoost but uses truncated exponential weights $q_t(x,y) = \min\{1, \exp(yF_{t-1}(x))\}$ instead of the logistic weights $q_t(x,y) = (1 + \exp(yF_t(x)))^{-1}$ used by FilterBoost. The algorithms' analyses differ, with MadaBoost requiring the edges $\gamma_t$ to be monotonically decreasing, but both lead to similar bounds on the number of rounds $T$ proportional to $\varepsilon^{-1}$. The non-adaptive filtering boosters of Freund [3,4] and of Bshouty & Gavinsky [6] and the batch booster AdaBoost [1] have smaller bounds on $T$, proportional to $\log(\varepsilon^{-1})$. However, we can use boosting tandems, a technique used by Freund [4] and Gavinsky [7], to create a filtering booster with $T$ bounded by $O(\log(\varepsilon^{-1})\gamma^{-2})$. Following Gavinsky, we can use FilterBoost to boost the accuracy of the weak learner to some constant and, in turn, treat FilterBoost as a weak learner and use an algorithm from Freund to achieve any target error. As with AdaFlat$_{filt}$, boosting tandems turn FilterBoost into an adaptive booster with a bound on $T$ proportional to $\log(\varepsilon^{-1})$. (Without boosting tandems, AdaFlat$_{filt}$ requires $T \propto \varepsilon^{-2}$ rounds.) Note, however, that boosting tandems result in more complicated final hypotheses.

An alternate bound for FilterBoost may be derived using techniques from Shalev-Shwartz & Singer [14]. They use the framework of convex repeated games to define a general method for bounding the performance of online and boosting algorithms. For FilterBoost, their techniques, combined with Theorem 2, give a bound similar to that in Theorem 3 but proportional to $\varepsilon^{-2}$ instead of $\varepsilon^{-1}$.

Schapire & Singer [9] show AdaBoost benefits from confidence-rated predictions, where weak hypotheses return predictions whose absolute values indicate confidence. These values are chosen to greedily minimize AdaBoost's exponential loss function over training data, and this aggressive weighting can result in faster learning. FilterBoost may use confidence-rated predictions in an identical manner. In the proof of Lemma 1, the decrease in the negative log likelihood $\pi_t$ of the data (relative to $H_t$ and the target distribution $D$) is lower-bounded by $p_t - p_t \sum_{(x,y)} D_t(x,y) e^{-\alpha_t y h_t(x)}$. Since $p_t$ is fixed, maximizing this bound is equivalent to minimizing the exponential loss over $D_t$.

# 3 Experiments

Vanilla FilterBoost accepts examples $(x,y)$ from the oracle with probability $q_t(x,y)$, but it may instead accept all examples and weight each with $q_t(x,y)$. Weighting instead of filtering examples increases accuracy but also increases the size of the training set passed to the weak learner. For efficiency, we choose to filter when training the weak learner but weight when estimating edges $\gamma_t$. We also modify FilterBoost's $getEdge()$ function for efficiency. The Nonmonotonic Adaptive Sampling (NAS) algorithm used to estimate edges $\gamma_t$ uses many examples, but using several orders of magnitude fewer sacrifices little accuracy. The same is true for MadaBoost. In all tests, we use $C_n \log(t+1)$ examples to estimate $\gamma_t$, where $C_n = 300$ and the log factor scales the number as the NAS algorithm would. For simplicity, we train weak learners with $C_n \log(t+1)$ examples as well. These modifications mean $\tau$ (error in edge estimates) and $\delta$ (confidence) have no effect on our tests. To simulate an oracle, we randomly permute the data and use examples in the new order. In practice, filtering boosters can achieve higher accuracy by cycling through training sets again instead of stopping once examples are depleted, and we use this "recycling" in our tests.

We test FilterBoost with and without confidence-rated predictions (labeled "(C-R)" in our results). We compare FilterBoost against MadaBoost [5], which does not require an a priori bound on weak hypotheses' edges and has similar bounds without the complication of boosting tandems. We implement MadaBoost with the same modifications as FilterBoost. We test FilterBoost against two batch boosters: the well-studied and historically successful AdaBoost [1] and the algorithm from Collins et al. [10] which is essentially a batch version of FilterBoost (labeled "AdaBoost-LOG"). We test both with and without confidence-rated predictions as well as with and without resampling (labeled "(resamp)"). In resampling, the booster trains weak learners on small sets of examples sampled from the distribution $D_t$ over the training set $S$ rather than on the entire set $S$, and this technique often increases efficiency with little effect on accuracy. Our batch boosters use sets of size $C_m \log(t+1)$ for training, like the filtering boosters, but use all of $S$ to estimate edges $\gamma_t$ since this can be done efficiently. We test the batch boosters using confidence-rated predictions and resampling in order to compare FilterBoost with batch algorithms optimized for the efficiency which boosting-by-filtering claims as its goal.

We test each booster using decision stumps and decision trees as weak learners to discern the effects of simple and complicated weak hypotheses. The decision stumps minimize training error, and the decision trees greedily maximize information gain and are pruned using $1/3$ of the data. Both weak learners minimize exponential loss when outputing confidence-rated predictions.

We use four datasets, described in the Appendix. Briefly, we use two synthetic sets: Majority (majority vote) and Twonorm [15], and two real sets from the UCI Machine Learning Repository [16]: Adult (census data; from Ron Kohavi) and Covertype (forestry data with 7 classes merged to 2; Copyr. Jock A. Blackard & Colorado State U.). We average over 10 runs, using new examples for synthetic data (with 50,000 test examples except where stated) and cross validation for real data.

Figure 2 compares the boosters' runtimes. As expected, filtering boosters run slower per round than batch boosters on small datasets but much faster on large ones. Interestingly, filtering boosters take longer on very small datasets in some cases (not shown), for the probability the filter accepts an example quickly shrinks when the booster has seen that example many times.

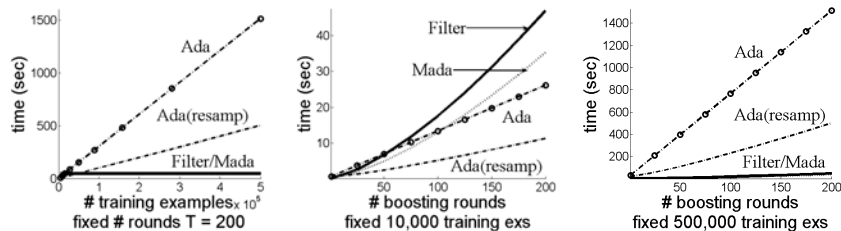

Figure 2: Running times: Ada/Filter/MadaBoost. Majority; WL = stumps.

## 3.1 Results: Conditional Probability Estimation

In Section 2.3, we discussed the interpretation of FilterBoost and Ada-Boost as stepwise algorithms for conditional probability estimation. We test both algorithms and the variants discussed above on all four datasets. We do not test MadaBoost, as it is not clear how to use it to estimate conditional probabilities. As Figure 3 shows, both FilterBoost variants are competitive with batch algorithms when boosting decision stumps. With decision trees, all algorithms except for FilterBoost overfit badly, including FilterBoost(C-R). In each plot, we compare FilterBoost with the best of AdaBoost and AdaBoost-LOG: AdaBoost was best with decision stumps and AdaBoost-LOG with decisions trees. For comparison, batch logistic regression via gradient descent achieves RMSE 0.3489 and log (base $e$) loss .4259; FilterBoost, inter-

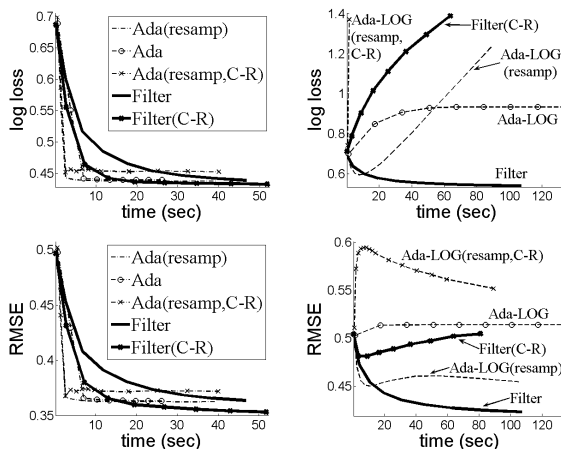

Figure 3: Log (base $e$) loss & root mean squared error (RMSE). Majority; 10,000 train exs.
Left two: WL = stumps (FilterBoost vs. AdaBoost);
Right two: WL = trees (FilterBoost vs. AdaBoost-LOG).

pretable as a stepwise method for logistic regression, seems to be approaching these asymptotically. On Adult and Twonorm, FilterBoost generally outperforms the batch boosters, which tend to overfit when boosting decision trees, though AdaBoost slightly outperforms FilterBoost on smaller datasets when boosting decision stumps.

The Covertype dataset is an exception to our results and highlights a danger in filtering and in resampling for batch learning: the complicated structure of some datasets seems to require decision trees to train on the entire dataset. With decision stumps, the filtering boosters are competitive, yet only the non-resampling batch boosters achieve high accuracies with decision trees. The first decision tree trained on the entire training set achieves about 94% accuracy, which is unachievable by any of the filtering or resampling batch boosters when using $C_m = 300$ as the base number of examples for training the weak learner. To compete with non-resampling batch boosters, the other boosters must use $C_m$ on the order of $10^5$, by which point they become very inefficient.

## 3.2 Results: Classification

Vanilla FilterBoost and MadaBoost perform similarly in classification (Figure 4). Confidence-rated predictions allow FilterBoost to outperform MadaBoost when using decision stumps but sometimes cause FilterBoost to perform poorly with decision trees. Figure 5 compares FilterBoost with the best batch booster for each weak learner. With decision stumps, all boosters achieve higher accuracies with the larger dataset, on which filtering algorithms are much more efficient. Majority is represented well as a linear combination of decision stumps, so the boosters all learn more slowly

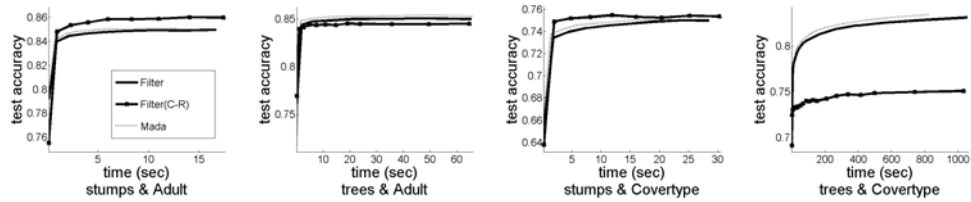

Figure 4: FilterBoost vs. MadaBoost.

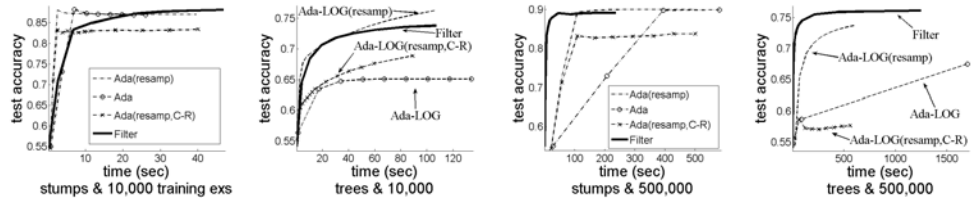

Figure 5: FilterBoost vs. AdaBoost & AdaBoost-LOG. Majority.

when using the overly complicated decision trees. However, this problem generally affects filtering boosters less than most batch variants, especially on larger datasets. Adult and Twonorm gave similar results. As in Section 3.1, filtering and resampling batch boosters perform poorly on Covertype. Thus, while FilterBoost is competitive in classification, its best performance is in regression.

### References

[1] Freund, Y., & Schapire, R. E. (1997) A decision-theoretic generalization of on-line learning and an application to boosting. *Journal of Computer and System Sciences, 55,* 119-139.

[2] Schapire., R. E. (1990) The strength of weak learnability. *Machine Learning, 5(2),* pp. 197-227.

[3] Freund, Y. (1995) Boosting a weak learning algorithm by majority. *Information and Computation, 121,* pp. 256-285.

[4] Freund, Y. (1992) An improved boosting algorithm and its implications on learning complexity. $5^{th}$ *Annual Conference on Computational Learning Theory,* pp. 391-398.

[5] Domingo, C., & Watanabe, O. (2000) MadaBoost: a modification of AdaBoost. $13^{th}$ *Annual Conference on Computational Learning Theory,* pp. 180-189.

[6] Bshouty, N. H., & Gavinsky, D. (2002) On boosting with polynomially bounded distributions. *Journal of Machine Learning Research, 3,* pp. 483-506.

[7] Gavinsky, D. (2003) Optimally-smooth adaptive boosting and application to agnostic learning. *Journal of Machine Learning Research, 4,* pp. 101-117.

[8] Hatano, K. (2006) Smooth boosting using an information-based criterion. $17^{th}$ *International Conference on Algorithmic Learning Theory,* pp. 304-319.

[9] Schapire, R. E., & Singer, Y. (1999) Improved boosting algorithms using confidence-rated predictions. *Machine Learning, 37,* 297-336.

[10] Collins, M., Schapire, R. E., & Singer, Y. (2002) Logistic regression, AdaBoost and Bregman distances. *Machine Learning, 48,* pp. 253-285.

[11] Watanabe, O. (2000) Simple sampling techniques for discovery science. *IEICE Trans. Information and Systems, E83-D(1),* 19-26.

[12] Domingo, C., Galvadà, R., & Watanabe, O. (2002) Adaptive sampling methods for scaling up knowledge discovery algorithms. *Data Mining and Knowledge Discovery, 6,* pp. 131-152.

[13] Friedman, J., Hastie, T., & Tibshirani, R. (2000) Additive logistic regression: a statistical view of boosting. *The Annals of Statistics, 28,* 337-407.

[14] Shalev-Shwartz, S., & Singer, Y. (2006) Convex repeated games and Fenchel duality. *Advances in Neural Information Processing Systems 20.*

[15] Breiman, L. (1998) Arcing classifiers. *The Annals of Statistics, 26,* pp. 801-849.

[16] Newman, D. J., Hettich, S., Blake, C. L., & Merz, C. J. (1998) UCI Repository of machine learning databases [http://www.ics.uci.edu/~mlearn/MLRepository.html]. Irvine, CA: U. of California, Dept. of Information & Computer Science.

